# Speedy Q-Learning

**Mohammad Gheshlaghi Azar**
Radboud University Nijmegen
Geert Grooteplein 21N, 6525 EZ
Nijmegen, Netherlands
m.azar@science.ru.nl

**Remi Munos**
INRIA Lille, SequeL Project
40 avenue Halley
59650 Villeneuve d'Ascq, France
r.munos@inria.fr

**Mohammad Ghavamzadeh**
INRIA Lille, SequeL Project
40 avenue Halley
59650 Villeneuve d'Ascq, France
m.ghavamzadeh@inria.fr

**Hilbert J. Kappen**
Radboud University Nijmegen
Geert Grooteplein 21N, 6525 EZ
Nijmegen, Netherlands
b.kappen@science.ru.nl

## Abstract

We introduce a new convergent variant of Q-learning, called speedy Q-learning (SQL), to address the problem of slow convergence in the standard form of the Q-learning algorithm. We prove a PAC bound on the performance of SQL, which shows that for an MDP with $n$ state-action pairs and the discount factor $\gamma$ only $T = O\big(\log(n)/(\epsilon^2(1-\gamma)^4)\big)$ steps are required for the SQL algorithm to converge to an $\epsilon$-optimal action-value function with high probability. This bound has a better dependency on $1/\epsilon$ and $1/(1-\gamma)$, and thus, is tighter than the best available result for Q-learning. Our bound is also superior to the existing results for both model-free and model-based instances of batch Q-value iteration that are considered to be more efficient than the incremental methods like Q-learning.

## 1 Introduction

Q-learning [20] is a well-known model-free reinforcement learning (RL) algorithm that finds an estimate of the optimal action-value function. Q-learning is a combination of dynamic programming, more specifically the value iteration algorithm, and stochastic approximation. In finite state-action problems, it has been shown that Q-learning converges to the optimal action-value function [5, 10]. However, it suffers from slow convergence, especially when the discount factor $\gamma$ is close to one [8, 17]. The main reason for the slow convergence of Q-learning is the combination of the sample-based stochastic approximation (that makes use of a decaying learning rate) and the fact that the Bellman operator propagates information throughout the whole space (specially when $\gamma$ is close to 1).

In this paper, we focus on RL problems that are formulated as finite state-action discounted infinite horizon Markov decision processes (MDPs), and propose an algorithm, called *speedy Q-learning* (SQL), that addresses the problem of slow convergence of Q-learning. At each time step, SQL uses two successive estimates of the action-value function that makes its space complexity twice as the standard Q-learning. However, this allows SQL to use a more aggressive learning rate for one of the terms in its update rule and eventually achieves a faster convergence rate than the standard Q-learning (see Section 3.1 for a more detailed discussion). We prove a PAC bound on the performance of SQL, which shows that only $T = O\big(\log(n)/((1-\gamma)^4\epsilon^2)\big)$ number of samples are required for SQL in order to guarantee an $\epsilon$-optimal action-value function with high probability. This is superior to the best result for the standard Q-learning by [8], both in terms of $1/\epsilon$ and $1/(1-\gamma)$. The rate for SQL is even better than that for the *Phased Q-learning* algorithm, a model-free batch Q-value

iteration algorithm proposed and analyzed by [12]. In addition, SQL's rate is slightly better than the rate of the model-based batch Q-value iteration algorithm in [12] and has a better computational and memory requirement (computational and space complexity), see Section 3.3.2 for more detailed comparisons. Similar to Q-learning, SQL may be implemented in synchronous and asynchronous fashions. For the sake of simplicity in the analysis, we only report and analyze its synchronous version in this paper. However, it can easily be implemented in an asynchronous fashion and our theoretical results can also be extended to this setting by following the same path as [8].

The idea of using previous estimates of the action-values has already been used to improve the performance of Q-learning. A popular algorithm of this kind is Q($\lambda$) [14, 20], which incorporates the concept of eligibility traces in Q-learning, and has been empirically shown to have a better performance than Q-learning, i.e., Q(0), for suitable values of $\lambda$. Another recent work in this direction is *Double Q-learning* [19], which uses two estimators for the action-value function to alleviate the over-estimation of action-values in Q-learning. This over-estimation is caused by a positive bias introduced by using the maximum action value as an approximation for the expected action value [19].

The rest of the paper is organized as follows. After introducing the notations used in the paper in Section 2, we present our *Speedy Q-learning* algorithm in Section 3. We first describe the algorithm in Section 3.1, then state our main theoretical result, i.e., a high-probability bound on the performance of SQL, in Section 3.2, and finally compare our bound with the previous results on Q-learning in Section 3.3. Section 4 contains the detailed proof of the performance bound of the SQL algorithm. Finally, we conclude the paper and discuss some future directions in Section 5.

## 2 Preliminaries

In this section, we introduce some concepts and definitions from the theory of Markov decision processes (MDPs) that are used throughout the paper. We start by the definition of supremum norm. For a real-valued function $g : \mathcal{Y} \mapsto \mathbb{R}$, where $\mathcal{Y}$ is a finite set, the supremum norm of $g$ is defined as $\|g\| \triangleq \max_{y \in \mathcal{Y}} |g(y)|$.

We consider the standard reinforcement learning (RL) framework [5, 16] in which a learning agent interacts with a stochastic environment and this interaction is modeled as a discrete-time discounted MDP. A discounted MDP is a quintuple $(\mathcal{X}, \mathcal{A}, P, \mathcal{R}, \gamma)$, where $\mathcal{X}$ and $\mathcal{A}$ are the set of states and actions, $P$ is the state transition distribution, $\mathcal{R}$ is the reward function, and $\gamma \in (0, 1)$ is a discount factor. We denote by $P(\cdot|x, a)$ and $r(x, a)$ the probability distribution over the next state and the immediate reward of taking action $a$ at state $x$, respectively. To keep the representation succinct, we use $\mathcal{Z}$ for the joint state-action space $\mathcal{X} \times \mathcal{A}$.

**Assumption 1** (MDP Regularity). *We assume $\mathcal{Z}$ and, subsequently, $\mathcal{X}$ and $\mathcal{A}$ are finite sets with cardinalities n, $|\mathcal{X}|$ and $|\mathcal{A}|$, respectively. We also assume that the immediate rewards $r(x, a)$ are uniformly bounded by $R_{\max}$ and define the horizon of the MDP $\beta \triangleq 1/(1-\gamma)$ and $V_{\max} \triangleq \beta R_{\max}$.*

A stationary Markov policy $\pi(\cdot|x)$ is the distribution over the control actions given the current state $x$. It is deterministic if this distribution concentrates over a single action. The *value* and the *action-value functions* of a policy $\pi$, denoted respectively by $V^{\pi} : \mathcal{X} \mapsto \mathbb{R}$ and $Q^{\pi} : \mathcal{Z} \mapsto \mathbb{R}$, are defined as the expected sum of discounted rewards that are encountered when the policy $\pi$ is executed. Given a MDP, the goal is to find a policy that attains the best possible values, $V^{*}(x) \triangleq \sup_{\pi} V^{\pi}(x), \ \forall x \in \mathcal{X}$. Function $V^{*}$ is called the *optimal value function*. Similarly the *optimal action-value function* is defined as $Q^{*}(x, a) = \sup_{\pi} Q^{\pi}(x, a), \ \forall(x, a) \in \mathcal{Z}$. The optimal action-value function $Q^{*}$ is the unique fixed-point of the *Bellman optimality operator* $\mathcal{T}$ defined as $(\mathcal{T}Q)(x, a) \triangleq r(x, a) + \gamma \sum_{y \in \mathcal{X}} P(y|x, a) \max_{b \in \mathcal{A}} Q(y, b), \ \forall(x, a) \in \mathcal{Z}$. It is important to note that $\mathcal{T}$ is a contraction with factor $\gamma$, i.e., for any pair of action-value functions $Q$ and $Q'$, we have $\|\mathcal{T}Q - \mathcal{T}Q'\| \leq \gamma \|Q - Q'\|$ [4, Chap. 1]. Finally for the sake of readability, we define the max operator $\mathcal{M}$ over action-value functions as $(\mathcal{M}Q)(x) = \max_{a \in \mathcal{A}} Q(x, a), \ \forall x \in \mathcal{X}$.

## 3 Speedy Q-Learning

In this section, we introduce our RL algorithm, called speedy Q-Learning (SQL), derive a performance bound for this algorithm, and compare this bound with similar results on standard Q-learning.

The derived performance bound shows that SQL has a rate of convergence of order $O(\sqrt{1/T})$, which is better than all the existing results for Q-learning.

## 3.1 Speedy Q-Learning Algorithm

The pseudo-code of the SQL algorithm is shown in Algorithm 1. As it can be seen, this is the synchronous version of the algorithm, which will be analyzed in the paper. Similar to the standard Q-learning, SQL may be implemented either synchronously or asynchronously. In the asynchronous version, at each time step, the action-value of the observed state-action pair is updated, while the rest of the state-action pairs remain unchanged. For the convergence of this instance of the algorithm, it is required that all the states and actions are visited infinitely many times, which makes the analysis slightly more complicated. On the other hand, given a generative model, the algorithm may be also formulated in a synchronous fashion, in which we first generate a next state $y \sim P(\cdot|x, a)$ for each state-action pair $(x, a)$, and then update the action-values of all the state-action pairs using these samples. We chose to include only the synchronous version of SQL in the paper just for the sake of simplicity in the analysis. However, the algorithm can be implemented in an asynchronous fashion (similar to the more familiar instance of Q-learning) and our theoretical results can also be extended to the asynchronous case under some mild assumptions.[1]

---

**Algorithm 1**: Synchronous Speedy Q-Learning (SQL)

**Input**: Initial action-value function $Q_0$, discount factor $\gamma$, and number of iteration $T$
$Q_{-1} := Q_0$;                                                    // Initialization
**for** $k := 0, 1, 2, 3, \dots, T-1$ **do**                        // Main loop
    $\alpha_k := \frac{1}{k+1}$;
    **for** *each* $(x, a) \in \mathcal{Z}$ **do**
        Generate the next state sample $y_k \sim P(\cdot|x, a)$;
        $\mathcal{T}_k Q_{k-1}(x, a) := r(x, a) + \gamma \mathcal{M} Q_{k-1}(y_k)$;
        $\mathcal{T}_k Q_k(x, a) := r(x, a) + \gamma \mathcal{M} Q_k(y_k)$;          // Empirical Bellman operator
        $Q_{k+1}(x, a) := Q_k(x, a) + \alpha_k\big(\mathcal{T}_k Q_{k-1}(x, a) - Q_k(x, a)\big) + (1-\alpha_k)\big(\mathcal{T}_k Q_k(x, a) - \mathcal{T}_k Q_{k-1}(x, a)\big)$;
                                                        // SQL update rule
    **end**
**end**
**return** $Q_T$

---

As it can be seen from Algorithm 1, at each time step $k$, SQL keeps track of the action-value functions of the two time-steps $k$ and $k-1$, and its main update rule is of the following form:

$$Q_{k+1}(x, a) = Q_k(x, a) + \alpha_k\big(\mathcal{T}_k Q_{k-1}(x, a) - Q_k(x, a)\big) + (1-\alpha_k)\big(\mathcal{T}_k Q_k(x, a) - \mathcal{T}_k Q_{k-1}(x, a)\big), \tag{1}$$

where $\mathcal{T}_k Q(x, a) = r(x, a) + \gamma \mathcal{M} Q(y_k)$ is the empirical Bellman optimality operator for the sampled next state $y_k \sim P(\cdot|x, a)$. At each time step $k$ and for state-action pair $(x, a)$, SQL works as follows: **(i)** it generates a next state $y_k$ by drawing a sample from $P(\cdot|x, a)$, **(ii)** it calculates two sample estimates $\mathcal{T}_k Q_{k-1}(x, a)$ and $\mathcal{T}_k Q_k(x, a)$ of the Bellman optimality operator (for state-action pair $(x, a)$ using the next state $y_k$) applied to the estimates $Q_{k-1}$ and $Q_k$ of the action-value function at the previous and current time steps, and finally **(iii)** it updates the action-value function of $(x, a)$, generates $Q_{k+1}(x, a)$, using the update rule of Eq. 1. Moreover, we let $\alpha_k$ decays linearly with time, i.e., $\alpha_k = 1/(k+1)$, in the SQL algorithm. [2]The update rule of Eq. 1 may be rewritten in the following more compact form:

$$Q_{k+1}(x, a) = (1 - \alpha_k)Q_k(x, a) + \alpha_k \mathcal{D}_k[Q_k, Q_{k-1}](x, a), \tag{2}$$

where $\mathcal{D}_k[Q_k, Q_{k-1}](x, a) \triangleq k\mathcal{T}_k Q_k(x, a) - (k-1)\mathcal{T}_k Q_{k-1}(x, a)$. This compact form will come specifically handy in the analysis of the algorithm in Section 4.

Let us consider the update rule of Q-learning

$$Q_{k+1}(x, a) = Q_k(x, a) + \alpha_k\big(\mathcal{T}_k Q_k(x, a) - Q_k(x, a)\big),$$

which may be rewritten as

$$Q_{k+1}(x,a) = Q_k(x,a) + \alpha_k \big( \mathcal{T}_k Q_{k-1}(x,a) - Q_k(x,a) \big) + \alpha_k \big( \mathcal{T}_k Q_k(x,a) - \mathcal{T}_k Q_{k-1}(x,a) \big). \quad (3)$$

Comparing the Q-learning update rule of Eq. 3 with the one for SQL in Eq. 1, we first notice that the same terms: $\mathcal{T}_k Q_{k-1} - Q_k$ and $\mathcal{T}_k Q_k - \mathcal{T}_k Q_{k-1}$ appear on the RHS of the update rules of both algorithms. However, while Q-learning uses the same conservative learning rate $\alpha_k$ for both these terms, SQL uses $\alpha_k$ for the first term and a bigger learning step $1 - \alpha_k = k/(k+1)$ for the second one. Since the term $\mathcal{T}_k Q_k - \mathcal{T}_k Q_{k-1}$ goes to zero as $Q_k$ approaches its optimal value $Q^*$, it is not necessary that its learning rate approaches zero. As a result, using the learning rate $\alpha_k$, which goes to zero with $k$, is too conservative for this term. This might be a reason why SQL that uses a more aggressive learning rate $1 - \alpha_k$ for this term has a faster convergence rate than Q-learning.

## 3.2 Main Theoretical Result

The main theoretical result of the paper is expressed as a high-probability bound over the performance of the SQL algorithm.

**Theorem 1.** *Let Assumption 1 holds and $T$ be a positive integer. Then, at iteration $T$ of SQL with probability at least $1 - \delta$, we have*

$$\|Q^* - Q_T\| \le 2\beta^2 R_{\max} \left[ \frac{\gamma}{T} + \sqrt{\frac{2 \log \frac{2n}{\delta}}{T}} \right].$$

We report the proof of Theorem 1 in Section 4. This result, combined with Borel-Cantelli lemma [9], guarantees that $Q_T$ converges almost surely to $Q^*$ with the rate $\sqrt{1/T}$. Further, the following result which quantifies the number of steps $T$ required to reach the error $\epsilon > 0$ in estimating the optimal action-value function, w.p. $1 - \delta$, is an immediate consequence of Theorem 1.

**Corollary 1** (Finite-time PAC ("probably approximately correct") performance bound for SQL). *Under Assumption 1, for any $\epsilon > 0$, after*

$$T = \frac{11.66 \beta^4 R_{\max}^2 \log \frac{2n}{\delta}}{\epsilon^2}$$

*steps of SQL, the uniform approximation error $\|Q^* - Q_T\| \le \epsilon$, with probability at least $1 - \delta$.*

## 3.3 Relation to Existing Results

In this section, we first compare our results for SQL with the existing results on the convergence of standard Q-learning. This comparison indicates that SQL accelerates the convergence of Q-learning, especially for $\gamma$ close to 1 and small $\epsilon$. We then compare SQL with batch Q-value iteration (QI) in terms of sample and computational complexities, i.e., the number of samples and the computational cost required to achieve an $\epsilon$-optimal solution w.p. $1 - \delta$, as well as space complexity, i.e., the memory required at each step of the algorithm.

### 3.3.1 A Comparison with the Convergence Rate of Standard Q-Learning

There are not many studies in the literature concerning the convergence rate of incremental model-free RL algorithms such as Q-learning. [17] has provided the asymptotic convergence rate for Q-learning under the assumption that all the states have the same next state distribution. This result shows that the asymptotic convergence rate of Q-learning has exponential dependency on $1 - \gamma$, i.e., the rate of convergence is of $\tilde{O}(1/t^{1-\gamma})$ for $\gamma \ge 1/2$.

The finite time behavior of Q-learning have been throughly investigated in [8] for different time scales. Their main result indicates that by using the polynomial learning step $\alpha_k = 1/(k+1)^\omega$, $0.5 < \omega < 1$, Q-learning achieves $\epsilon$-optimal performance w.p. at least $1 - \delta$ after

$$T = O \left( \left[ \frac{\beta^4 R_{\max}^2 \log \frac{n \beta R_{\max}}{\delta \epsilon}}{\epsilon^2} \right]^{\frac{1}{\omega}} + \left[ \beta \log \frac{\beta R_{\max}}{\epsilon} \right]^{\frac{1}{1-\omega}} \right) \quad (4)$$

steps. When $\gamma \approx 1$, one can argue that $\beta = 1/(1 - \gamma)$ becomes the dominant term in the bound of Eq. 4, and thus, the optimized bound w.r.t. $\omega$ is obtained for $\omega = 4/5$ and is of $\tilde{O}\big(\beta^5/\epsilon^{2.5}\big)$. On the other hand, SQL is guaranteed to achieve the same precision in only $O\big(\beta^4/\epsilon^2\big)$ steps. The difference between these two bounds is significant for large values of $\beta$, i.e., $\gamma$'s close to 1.

### 3.3.2 SQL vs. Q-Value Iteration

Finite sample bounds for both model-based and model-free (Phased Q-learning) QI have been derived in [12] and [7]. These algorithms can be considered as the batch version of Q-learning. They show that to quantify $\epsilon$-optimal action-value functions with high probability, we need $O\big(n\beta^5/\epsilon^2 \log(1/\epsilon)\big(\log(n\beta) + \log\log 1/\epsilon\big)\big)$ and $O\big(n\beta^4/\epsilon^2(\log(n\beta) + \log\log 1/\epsilon)\big)$ samples in model-free and model-based QI, respectively. A comparison between their results and the main result of this paper suggests that the sample complexity of SQL, which is of order $O\big(n\beta^4/\epsilon^2 \log n\big)$,[3] is better than model-free QI in terms of $\beta$ and $\log(1/\epsilon)$. Although the sample complexities of SQL is only slightly tighter than the model-based QI, SQL has a significantly better computational and space complexity than model-based QI: SQL needs only $2n$ memory space, while the space complexity of model-based QI is given by either $\tilde{O}(n\beta^4/\epsilon^2)$ or $n(|\mathcal{X}| + 1)$, depending on whether the learned state transition matrix is sparse or not [12]. Also, SQL improves the computational complexity by a factor of $\tilde{O}(\beta)$ compared to both model-free and model-based QI.[4] Table 1 summarizes the comparisons between SQL and the other RL methods discussed in this section.

Table 1: Comparison between SQL, Q-learning, model-based and model-free Q-value iteration in terms of sample complexity (SC), computational complexity (CC), and space complexity (SPC).

| Method | SQL | Q-learning (optimized) | Model-based QI | Model-free QI |
|---|---|---|---|---|
| SC | $\tilde{O}\left(\dfrac{n\beta^4}{\epsilon^2}\right)$ | $\tilde{O}\left(\dfrac{n\beta^5}{\epsilon^{2.5}}\right)$ | $\tilde{O}\left(\dfrac{n\beta^4}{\epsilon^2}\right)$ | $\tilde{O}\left(\dfrac{n\beta^5}{\epsilon^2}\right)$ |
| CC | $\tilde{O}\left(\dfrac{n\beta^4}{\epsilon^2}\right)$ | $\tilde{O}\left(\dfrac{n\beta^5}{\epsilon^{2.5}}\right)$ | $\tilde{O}\left(\dfrac{n\beta^5}{\epsilon^2}\right)$ | $\tilde{O}\left(\dfrac{n\beta^5}{\epsilon^2}\right)$ |
| SPC | $\Theta(n)$ | $\Theta(n)$ | $\tilde{O}\left(\dfrac{n\beta^4}{\epsilon^2}\right)$ | $\Theta(n)$ |

## 4   Analysis

In this section, we give some intuition about the convergence of SQL and provide the full proof of the finite-time analysis reported in Theorem 1. We start by introducing some notations.

Let $\mathcal{F}_k$ be the filtration generated by the sequence of all random samples $\{y_1, y_2, \ldots, y_k\}$ drawn from the distribution $P(\cdot|x, a)$, for all state action $(x, a)$ up to round $k$. We define the operator $\mathcal{D}[Q_k, Q_{k-1}]$ as the expected value of the empirical operator $\mathcal{D}_k$ conditioned on $\mathcal{F}_{k-1}$:

$$\mathcal{D}[Q_k, Q_{k-1}](x, a) \triangleq \mathbb{E}(\mathcal{D}_k[Q_k, Q_{k-1}](x, a)\,|\mathcal{F}_{k-1})$$
$$= k\mathcal{T}Q_k(x, a) - (k-1)\mathcal{T}Q_{k-1}(x, a).$$

Thus the update rule of SQL writes

$$Q_{k+1}(x, a) = (1 - \alpha_k)Q_k(x, a) + \alpha_k\left(\mathcal{D}[Q_k, Q_{k-1}](x, a) - \epsilon_k(x, a)\right), \qquad (5)$$

where the estimation error $\epsilon_k$ is defined as the difference between the operator $\mathcal{D}[Q_k, Q_{k-1}]$ and its sample estimate $\mathcal{D}_k[Q_k, Q_{k-1}]$ for all $(x, a) \in \mathcal{Z}$:

$$\epsilon_k(x, a) \triangleq \mathcal{D}[Q_k, Q_{k-1}](x, a) - \mathcal{D}_k[Q_k, Q_{k-1}](x, a).$$

We have the property that $\mathbb{E}[\epsilon_k(x, a)|\mathcal{F}_{k-1}] = 0$ which means that for all $(x, a) \in \mathcal{Z}$ the sequence of estimation error $\{\epsilon_1(x, a), \epsilon_2(x, a), \ldots, \epsilon_k(x, a)\}$ is a martingale difference sequence w.r.t. the filtration $\mathcal{F}_k$. Let us define the martingale $E_k(x, a)$ to be the sum of the estimation errors:

$$E_k(x, a) \triangleq \sum_{j=0}^{k} \epsilon_j(x, a), \qquad\qquad \forall(x, a) \in \mathcal{Z}. \qquad (6)$$

The proof of Theorem 1 follows the following steps: **(i)** Lemma 1 shows the stability of the algorithm (i.e., the sequence of $Q_k$ stays bounded). **(ii)** Lemma 2 states the key property that the SQL iterate $Q_{k+1}$ is very close to the Bellman operator $\mathcal{T}$ applied to the previous iterate $Q_k$ plus an estimation error term of order $E_k/k$. **(iii)** By induction, Lemma 3 provides a performance bound $\|Q^* - Q_k\|$ in terms of a discounted sum of the cumulative estimation errors $\{E_j\}_{j=0:k-1}$. Finally **(iv)** we use a maximal Azuma's inequality (see Lemma 4) to bound $E_k$ and deduce the finite time performance for SQL.

For simplicity of the notations, we remove the dependence on $(x, a)$ (e.g., writing $Q$ for $Q(x, a)$, $E_k$ for $E_k(x, a)$) when there is no possible confusion.

**Lemma 1** (Stability of SQL). *Let Assumption 1 hold and assume that the initial action-value function $Q_0 = Q_{-1}$ is uniformly bounded by $V_{\max}$, then we have, for all $k \geq 0$,*

$$\|Q_k\| \leq V_{\max}, \quad \|\epsilon_k\| \leq 2V_{\max}, \quad and \quad \|\mathcal{D}_k[Q_k, Q_{k-1}]\| \leq V_{\max}.$$

*Proof.* We first prove that $\|\mathcal{D}_k[Q_k, Q_{k-1}]\| \leq V_{\max}$ by induction. For $k = 0$ we have:

$$\|\mathcal{D}_0[Q_0, Q_{-1}]\| \leq \|r\| + \gamma\|\mathcal{M}Q_{-1}\| \leq R_{\max} + \gamma V_{\max} = V_{\max}.$$

Now for any $k \geq 0$, let us assume that the bound $\|\mathcal{D}_k[Q_k, Q_{k-1}]\| \leq V_{\max}$ holds. Thus

$$\|\mathcal{D}_{k+1}[Q_{k+1}, Q_k]\| \leq \|r\| + \gamma \|(k+1)\mathcal{M}Q_{k+1} - k\mathcal{M}Q_k\|$$

$$= \|r\| + \gamma \left\|(k+1)\mathcal{M}\left(\frac{k}{k+1}Q_k + \frac{1}{k+1}\mathcal{D}_k[Q_k, Q_{k-1}]\right) - k\mathcal{M}Q_k\right\|$$

$$\leq \|r\| + \gamma \|\mathcal{M}(kQ_k + \mathcal{D}_k[Q_k, Q_{k-1}] - kQ_k)\|$$

$$\leq \|r\| + \gamma \|\mathcal{D}_k[Q_k, Q_{k-1}]\| \leq R_{\max} + \gamma V_{\max} = V_{\max},$$

and by induction, we deduce that for all $k \geq 0$, $\|\mathcal{D}_k[Q_k, Q_{k-1}]\| \leq V_{\max}$.

Now the bound on $\epsilon_k$ follows from $\|\epsilon_k\| = \|\mathbb{E}(\mathcal{D}_k[Q_k, Q_{k-1}]|\mathcal{F}_{k-1}) - \mathcal{D}_k[Q_k, Q_{k-1}]\| \leq 2V_{\max}$, and the bound $\|Q_k\| \leq V_{\max}$ is deduced by noticing that $Q_k = 1/k \sum_{j=0}^{k-1} \mathcal{D}_j[Q_j, Q_{j-1}]$. $\quad\square$

The next lemma shows that $Q_k$ is close to $\mathcal{T}Q_{k-1}$, up to a $O(1/k)$ term plus the average cumulative estimation error $\frac{1}{k}E_{k-1}$.

**Lemma 2.** *Under Assumption 1, for any $k \geq 1$:*

$$Q_k = \frac{1}{k}\left(\mathcal{T}Q_0 + (k-1)\mathcal{T}Q_{k-1} - E_{k-1}\right). \qquad (7)$$

*Proof.* We prove this result by induction. The result holds for $k = 1$, where (7) reduces to (5). We now show that if the property (7) holds for $k$ then it also holds for $k + 1$. Assume that (7) holds for $k$. Then, from (5) we have:

$$Q_{k+1} = \frac{k}{k+1}Q_k + \frac{1}{k+1}(k\mathcal{T}Q_k - (k-1)\mathcal{T}Q_{k-1} - \epsilon_k)$$

$$= \frac{k}{k+1}\left(\frac{1}{k}(\mathcal{T}Q_0 + (k-1)\mathcal{T}Q_{k-1} - E_{k-1})\right) + \frac{1}{k+1}(k\mathcal{T}Q_k - (k-1)\mathcal{T}Q_{k-1} - \epsilon_k)$$

$$= \frac{1}{k+1}(\mathcal{T}Q_0 + k\mathcal{T}Q_k - E_{k-1} - \epsilon_k) = \frac{1}{k+1}(\mathcal{T}Q_0 + k\mathcal{T}Q_k - E_k).$$

Thus (7) holds for $k + 1$, and is thus true for all $k \geq 1$. $\quad\square$

Now we bound the difference between $Q^*$ and $Q_k$ in terms of the discounted sum of cumulative estimation errors $\{E_0, E_1, \ldots, E_{k-1}\}$.

**Lemma 3** (Error Propagation of SQL). *Let Assumption 1 hold and assume that the initial action-value function $Q_0 = Q_{-1}$ is uniformly bounded by $V_{\max}$, then for all $k \geq 1$, we have*

$$\|Q^* - Q_k\| \leq \frac{2\gamma\beta V_{\max}}{k} + \frac{1}{k}\sum_{j=1}^{k}\gamma^{k-j}\|E_{j-1}\|. \tag{8}$$

*Proof.* Again we prove this lemma by induction. The result holds for $k = 1$ as:
$$\|Q^* - Q_1\| = \|\mathfrak{T}Q^* - \mathfrak{T}_0 Q_0\| = \|\mathfrak{T}Q^* - \mathfrak{T}Q_0 + \epsilon_0\|$$
$$\leq \|\mathfrak{T}Q^* - \mathfrak{T}Q_0\| + \|\epsilon_0\| \leq 2\gamma V_{\max} + \|\epsilon_0\| \leq 2\gamma\beta V_{\max} + \|E_0\|$$
We now show that if the bound holds for $k$, then it also holds for $k + 1$. Thus, assume that (8) holds for $k$. By using Lemma 2:

$$\|Q^* - Q_{k+1}\| = \left\| Q^* - \frac{1}{k+1}(\mathfrak{T}Q_0 + k\mathfrak{T}Q_k - E_k) \right\|$$

$$= \left\| \frac{1}{k+1}(\mathfrak{T}Q^* - \mathfrak{T}Q_0) + \frac{k}{k+1}(\mathfrak{T}Q^* - \mathfrak{T}Q_k) + \frac{1}{k+1}E_k \right\|$$

$$\leq \frac{\gamma}{k+1}\|Q^* - Q_0\| + \frac{k\gamma}{k+1}\|Q^* - Q_k\| + \frac{1}{k+1}\|E_k\|$$

$$\leq \frac{2\gamma}{k+1}V_{\max} + \frac{k\gamma}{k+1}\left[ \frac{2\gamma\beta V_{\max}}{k} + \frac{1}{k}\sum_{j=1}^{k}\gamma^{k-j}\|E_{j-1}\| \right] + \frac{1}{k+1}\|E_k\|$$

$$= \frac{2\gamma\beta V_{\max}}{k+1} + \frac{1}{k+1}\sum_{j=1}^{k+1}\gamma^{k+1-j}\|E_{j-1}\|.$$

Thus (8) holds for $k + 1$ thus for all $k \geq 1$ by induction. $\qquad\square$

Now, based on Lemmas 3 and 1, we prove the main theorem of this paper.

***Proof of Theorem 1.*** We begin our analysis by recalling the result of Lemma 3 at round $T$:

$$\|Q^* - Q_T\| \leq \frac{2\gamma\beta V_{\max}}{T} + \frac{1}{T}\sum_{k=1}^{T}\gamma^{T-k}\|E_{k-1}\|.$$

Note that the difference between this bound and the result of Theorem 1 is just in the second term. So, we only need to show that the following inequality holds, with probability at least $1 - \delta$:

$$\frac{1}{T}\sum_{k=1}^{T}\gamma^{T-k}\|E_{k-1}\| \leq 2\beta V_{\max}\sqrt{\frac{2\log\frac{2n}{\delta}}{T}}. \tag{9}$$

We first notice that:

$$\frac{1}{T}\sum_{k=1}^{T}\gamma^{T-k}\|E_{k-1}\| \leq \frac{1}{T}\sum_{k=1}^{T}\gamma^{T-k}\max_{1\leq k\leq T}\|E_{k-1}\| \leq \frac{\beta\max_{1\leq k\leq T}\|E_{k-1}\|}{T}. \tag{10}$$

Therefore, in order to prove (9) it is sufficient to bound $\max_{1\leq k\leq T}\|E_{k-1}\| = \max_{(x,a)\in\mathcal{Z}}\max_{1\leq k\leq T}|E_{k-1}(x,a)|$ in high probability. We start by providing a high probability bound for $\max_{1\leq k\leq T}|E_{k-1}(x,a)|$ for a given $(x,a)$. First notice that

$$\mathbb{P}\left( \max_{1\leq k\leq T}|E_{k-1}(x,a)| > \epsilon \right) = \mathbb{P}\left( \max\left[ \max_{1\leq k\leq T}(E_{k-1}(x,a)), \max_{1\leq k\leq T}(-E_{k-1}(x,a)) \right] > \epsilon \right)$$

$$= \mathbb{P}\left( \left\{ \max_{1\leq k\leq T}(E_{k-1}(x,a)) > \epsilon \right\} \bigcup \left\{ \max_{1\leq k\leq T}(-E_{k-1}(x,a)) > \epsilon \right\} \right)$$

$$\leq \mathbb{P}\left( \max_{1\leq k\leq T}(E_{k-1}(x,a)) > \epsilon \right) + \mathbb{P}\left( \max_{1\leq k\leq T}(-E_{k-1}(x,a)) > \epsilon \right), \tag{11}$$

and each term is now bounded by using a maximal Azuma inequality, reminded now (see e.g., [6]).

**Lemma 4** (Maximal Hoeffding-Azuma Inequality). *Let $\mathcal{V} = \{V_1, V_2, \ldots, V_T\}$ be a martingale difference sequence w.r.t. a sequence of random variables $\{X_1, X_2, \ldots, X_T\}$ (i.e., $\mathbb{E}(V_{k+1}|X_1, \ldots X_k) = 0$ for all $0 < k \leq T$) such that $\mathcal{V}$ is uniformly bounded by $L > 0$. If we define $S_k = \sum_{i=1}^k V_i$, then for any $\epsilon > 0$, we have*

$$\mathbb{P}\left(\max_{1 \leq k \leq T} S_k > \epsilon\right) \leq \exp\left(\frac{-\epsilon^2}{2TL^2}\right).$$

As mentioned earlier, the sequence of random variables $\{\epsilon_0(x,a), \epsilon_1(x,a), \cdots, \epsilon_k(x,a)\}$ is a martingale difference sequence w.r.t. the filtration $\mathcal{F}_k$ (generated by the random samples $\{y_0, y_1, \ldots, y_k\}(x,a)$ for all $(x,a)$), i.e., $\mathbb{E}[\epsilon_k(x,a)|\mathcal{F}_{k-1}] = 0$. It follows from Lemma 4 that for any $\epsilon > 0$ we have:

$$
\begin{aligned}
\mathbb{P}\left(\max_{1 \leq k \leq T}(E_{k-1}(x,a)) > \epsilon\right) &\leq \exp\left(\frac{-\epsilon^2}{8TV_{\max}^2}\right) \\
\mathbb{P}\left(\max_{1 \leq k \leq T}(-E_{k-1}(x,a)) > \epsilon\right) &\leq \exp\left(\frac{-\epsilon^2}{8TV_{\max}^2}\right).
\end{aligned}
\tag{12}
$$

By combining (12) with (11) we deduce that $\mathbb{P}\left(\max_{1 \leq k \leq T}|E_{k-1}(x,a)| > \epsilon\right) \leq 2\exp\left(\frac{-\epsilon^2}{8TV_{\max}^2}\right)$, and by a union bound over the state-action space, we deduce that

$$\mathbb{P}\left(\max_{1 \leq k \leq T}\|E_{k-1}\| > \epsilon\right) \leq 2n\exp\left(\frac{-\epsilon^2}{8TV_{\max}^2}\right). \tag{13}$$

This bound can be rewritten as: for any $\delta > 0$,

$$\mathbb{P}\left(\max_{1 \leq k \leq T}\|E_{k-1}\| \leq V_{\max}\sqrt{8T\log\frac{2n}{\delta}}\right) \geq 1 - \delta, \tag{14}$$

which by using (10) proves (9) and Theorem 1. □

## 5 Conclusions and Future Work

In this paper, we introduced a new Q-learning algorithm, called speedy Q-learning (SQL). We analyzed the finite time behavior of this algorithm as well as its asymptotic convergence to the optimal action-value function. Our result is in the form of high probability bound on the performance loss of SQL, which suggests that the algorithm converges to the optimal action-value function in a faster rate than the standard Q-learning. Overall, SQL is a simple, efficient and theoretically well-founded reinforcement learning algorithm, which improves on existing RL algorithms such as Q-learning and model-based value iteration.

In this work, we are only interested in the estimation of the optimal action-value function and not the problem of exploration. Therefore, we did not compare our result to the PAC-MDP methods [15, 18] and the upper-confidence bound based algorithms [3, 11], in which the choice of the exploration policy impacts the behavior of the learning algorithms. However, we believe that it would be possible to gain w.r.t. the state of the art in PAC-MDPs, by combining the asynchronous version of SQL with a smart exploration strategy. This is mainly due to the fact that the bound for SQL has been proved to be tighter than the RL algorithms that have been used for estimating the value function in PAC-MDP methods, especially in the model-free case. We consider this as a subject for future research.

Another possible direction for future work is to scale up SQL to large (possibly continuous) state and action spaces where function approximation is needed. We believe that it would be possible to extend our current SQL analysis to the continuous case along the same path as in the fitted value iteration analysis by [13] and [1]. This would require extending the error propagation result of Lemma 3 to a $\ell_2$-norm analysis and combining it with the standard regression bounds.

**Acknowledgments**

The authors appreciate supports from the PASCAL2 Network of Excellence Internal-Visit Programme and the European Community's Seventh Framework Programme (FP7/2007-2013) under grant agreement n° 231495. We also thank Peter Auer for helpful discussion and the anonymous reviewers for their valuable comments.

## Footnotes

[1]See [2] for the convergence analysis of the asynchronous variant of SQL.

[2]Note that other (polynomial) learning steps can also be used with speedy Q-learning. However one can show that the rate of convergence of SQL is optimized for $\alpha_k = 1/(k+1)$. This is in contrast to the standard Q-learning algorithm for which the rate of convergence is optimized for a polynomial learning step [8].

[3]Note that at each round of SQL $n$ new samples are generated. This combined with the result of Corollary 1 deduces the sample complexity of order $O(n\beta^4/\epsilon^2 \log(n/\delta))$.

[4]SQL has the sample and computational complexity of a same order since it performs only one Q-value update per sample, whereas, in the case of model-based QI, the algorithm needs to iterate the action-value function of all state-action pairs at least $\tilde{O}(\beta)$ times using Bellman operator, which leads to a computational complexity bound of order $\tilde{O}(n\beta^5/\epsilon^2)$ given that only $\tilde{O}(n\beta^4/\epsilon^2)$ entries of the estimated transition matrix are non-zero [12].

# References

[1] A. Antos, R. Munos, and Cs. Szepesvári. Fitted Q-iteration in continuous action-space MDPs. In *Proceedings of the 21st Annual Conference on Neural Information Processing Systems*, 2007.

[2] M. Gheshlaghi Azar, R. Munos, M. Ghavamzadeh, and H.J. Kappen. Reinforcement learning with a near optimal rate of convergence. Technical Report inria-00636615, INRIA, 2011.

[3] P. L. Bartlett and A. Tewari. REGAL: A regularization based algorithm for reinforcement learning in weakly communicating MDPs. In *Proceedings of the 25th Conference on Uncertainty in Artificial Intelligence*, 2009.

[4] D. P. Bertsekas. *Dynamic Programming and Optimal Control*, volume II. Athena Scientific, Belmount, Massachusetts, third edition, 2007.

[5] D. P. Bertsekas and J. N. Tsitsiklis. *Neuro-Dynamic Programming*. Athena Scientific, Belmont, Massachusetts, 1996.

[6] N. Cesa-Bianchi and G. Lugosi. *Prediction, Learning, and Games*. Cambridge University Press, New York, NY, USA, 2006.

[7] E. Even-Dar, S. Mannor, and Y. Mansour. PAC bounds for multi-armed bandit and Markov decision processes. In *15th Annual Conference on Computational Learning Theory*, pages 255–270, 2002.

[8] E. Even-Dar and Y. Mansour. Learning rates for Q-learning. *Journal of Machine Learning Research*, 5:1–25, 2003.

[9] W. Feller. *An Introduction to Probability Theory and Its Applications*, volume 1. Wiley, 1968.

[10] T. Jaakkola, M. I. Jordan, and S. Singh. On the convergence of stochastic iterative dynamic programming. *Neural Computation*, 6(6):1185–1201, 1994.

[11] T. Jaksch, R. Ortner, and P. Auer. Near-optimal regret bounds for reinforcement learning. *Journal of Machine Learning Research*, 11:1563–1600, 2010.

[12] M. Kearns and S. Singh. Finite-sample convergence rates for Q-learning and indirect algorithms. In *Advances in Neural Information Processing Systems 12*, pages 996–1002. MIT Press, 1999.

[13] R. Munos and Cs. Szepesvári. Finite-time bounds for fitted value iteration. *Journal of Machine Learning Research*, 9:815–857, 2008.

[14] J. Peng and R. J. Williams. Incremental multi-step Q-learning. *Machine Learning*, 22(1-3):283–290, 1996.

[15] A. L. Strehl, L. Li, and M. L. Littman. Reinforcement learning in finite MDPs: PAC analysis. *Journal of Machine Learning Research*, 10:2413–2444, 2009.

[16] R. S. Sutton and A. G. Barto. *Reinforcement Learning: An Introduction*. MIT Press, Cambridge, Massachusetts, 1998.

[17] Cs. Szepesvári. The asymptotic convergence-rate of Q-learning. In *Advances in Neural Information Processing Systems 10, Denver, Colorado, USA, 1997*, 1997.

[18] I. Szita and Cs. Szepesvári. Model-based reinforcement learning with nearly tight exploration complexity bounds. In *Proceedings of the 27th International Conference on Machine Learning*, pages 1031–1038. Omnipress, 2010.

[19] H. van Hasselt. Double Q-learning. In *Advances in Neural Information Processing Systems 23*, pages 2613–2621, 2010.

[20] C. Watkins. *Learning from Delayed Rewards*. PhD thesis, Kings College, Cambridge, England, 1989.

